# Breaking Audio CAPTCHAs

**Jennifer Tam**
Computer Science Department
Carnegie Mellon University
5000 Forbes Ave, Pittsburgh 15217
*jdtam@cs.cmu.edu*

**Jiri Simsa**
Computer Science Department
Carnegie Mellon University
5000 Forbes Ave, Pittsburgh 15217
*jsimsa@cs.cmu.edu*

**Sean Hyde**
Electrical and Computer Engineering
Carnegie Mellon University
5000 Forbes Ave, Pittsburgh 15217
*sean.a.hyde@gmail.com*

**Luis Von Ahn**
Computer Science Department
Carnegie Mellon University
5000 Forbes Ave, Pittsburgh 15217
*biglou@cs.cmu.edu*

## *Abstract*

CAPTCHAs are computer-generated tests that humans can pass but current computer systems cannot. CAPTCHAs provide a method for automatically distinguishing a human from a computer program, and therefore can protect Web services from abuse by so-called "bots." Most CAPTCHAs consist of distorted images, usually text, for which a user must provide some description. Unfortunately, visual CAPTCHAs limit access to the millions of visually impaired people using the Web. Audio CAPTCHAs were created to solve this accessibility issue; however, the security of audio CAPTCHAs was never formally tested. Some visual CAPTCHAs have been broken using machine learning techniques, and we propose using similar ideas to test the security of audio CAPTCHAs. Audio CAPTCHAs are generally composed of a set of words to be identified, layered on top of noise. We analyzed the security of current audio CAPTCHAs from popular Web sites by using AdaBoost, SVM, and k-NN, and achieved correct solutions for test samples with accuracy up to 71%. Such accuracy is enough to consider these CAPTCHAs broken. Training several different machine learning algorithms on different types of audio CAPTCHAs allowed us to analyze the strengths and weaknesses of the algorithms so that we could suggest a design for a more robust audio CAPTCHA.

## 1    Introduction

CAPTCHAs [1] are automated tests designed to tell computers and humans apart by presenting users with a problem that humans can solve but current computer programs cannot. Because CAPTCHAs can distinguish between humans and computers with high probability, they are used for many different security applications: they prevent bots from voting continuously in online polls, automatically registering for millions of spam email accounts, automatically purchasing tickets to buy out an event, etc. Once a CAPTCHA is broken (i.e., computer programs can successfully pass the test), bots can impersonate humans and gain access to services that they should not. Therefore, it is important for CAPTCHAs to be secure.

To pass the typical visual CAPTCHA, a user must correctly type the characters displayed in an image of distorted text. Many visual CAPTCHAs have been broken with machine

learning techniques [2]-[3], though some remain secure against such attacks. Because visually impaired users who surf the Web using screen-reading programs cannot see this type of CAPTCHA, audio CAPTCHAs were created. Typical audio CAPTCHAs consist of one or several speakers saying letters or digits at randomly spaced intervals. A user must correctly identify the digits or characters spoken in the audio file to pass the CAPTCHA. To make this test difficult for current computer systems, specifically automatic speech recognition (ASR) programs, background noise is injected into the audio files.

Since no official evaluation of existing audio CAPTCHAs has been reported, we tested the security of audio CAPTCHAs used by many popular Web sites by running machine learning experiments designed to break them. In the next section, we provide an overview of the literature related to our project. Section 3 describes our methods for creating training data, and section 4 describes how we create classifiers that can recognize letters, digits, and noise. In section 5, we discuss how we evaluated our methods on widely used audio CAPTCHAs and we give our results. In particular, we show that the audio CAPTCHAs used by sites such as Google and Digg are susceptible to machine learning attacks. Section 6 mentions the proposed design of a new more secure audio CAPTCHA based on our findings.

## 2    Literature review

To break the audio CAPTCHAs, we derive features from the CAPTCHA audio and use several machine learning techniques to perform ASR on segments of the CAPTCHA. There are many popular techniques for extracting features from speech. The three techniques we use are *mel-frequency cepstral coefficients* (MFCC), *perceptual linear prediction* (PLP), and *relative spectral transform-PLP* (RASTA-PLP). MFCC is one of the most popular speech feature representations used. Similar to a fast Fourier transform (FFT), MFCC transforms an audio file into frequency bands, but (unlike FFT) MFCC uses mel-frequency bands, which are better for approximating the range of frequencies humans hear. PLP was designed to extract speaker-independent features from speech [4]. Therefore, by using PLP and a variant such as RASTA-PLP, we were able to train our classifiers to recognize letters and digits independently of who spoke them. Since many different people recorded the digits used in one of the types of audio CAPTCHAs we tested, PLP and RASTA-PLP were needed to extract the features that were most useful for solving them.

In [4]-[5], the authors conducted experiments on recognizing isolated digits in the presence of noise using both PLP and RASTA-PLP. However, the noise used consisted of telephone or microphone static caused by recording in different locations. The audio CAPTCHAs we use contain this type of noise, as well as added vocal noise and/or music, which is supposed to make the automated recognition process much harder.

The authors of [3] emphasize how many visual CAPTCHAs can be broken by successfully splitting the task into two smaller tasks: segmentation and recognition. We follow a similar approach in that we first automatically split the audio into segments, and then we classify these segments as noise or words.

In early March 2008, concurrent to our work, the blog of Wintercore Labs [6] claimed to have successfully broken the Google audio CAPTCHA. After reading their Web article and viewing the video of how they solve the CAPTCHAs, we are unconvinced that the process is entirely automatic, and it is unclear what their exact pass rate is. Because we are unable to find any formal technical analysis of this program, we can neither be sure of its accuracy nor the extent of its automation.

## 3    Creation of training data

Since automated programs can attempt to pass a CAPTCHA repeatedly, a CAPTCHA is essentially broken when a program can pass it more than a non-trivial fraction of the time; e.g., a 5% pass rate is enough.

Our approach to breaking the audio CAPTCHAs began by first splitting the audio files into segments of noise or words: for our experiments, the words were spoken letters or digits. We used manual transcriptions of the audio CAPTCHAs to get information regarding the location of each spoken word within the audio file. We were able to label our segments accurately by using this information.

We gathered 1,000 audio CAPTCHAs from each of the following Web sites: google.com, digg.com, and an older version of the audio CAPTCHA in recaptcha.net. Each of the CAPTCHAs was annotated with the information regarding letter/digit locations provided by the manual transcriptions. For each type of CAPTCHA, we randomly selected 900 samples for training and used the remaining 100 for testing.

Using the digit/letter location information provided in the manual CAPTCHA transcriptions, each training CAPTCHA is divided into segments of noise, the letters a-z, or the digits 0-9, and labeled as such. We ignore the annotation information of the CAPTCHAs we use for testing, and therefore we cannot identify the size of those segments. Instead, each test CAPTCHA is divided into a number of fixed-size segments. The segments with the highest energy peaks are then classified using machine learning techniques (Figure 1). Since the size of a feature vector extracted from a segment generally depends on the size of the segment, using fixed-size segments allows each segment to be described with a feature vector of the same length. We chose the window size by listening to a few training segments and adjusted accordingly to ensure that the segment contained the entire digit/letter. There is undoubtedly a more optimal way of selecting the window size, however, we were still able to break the three CAPTCHAs we tested with our method.

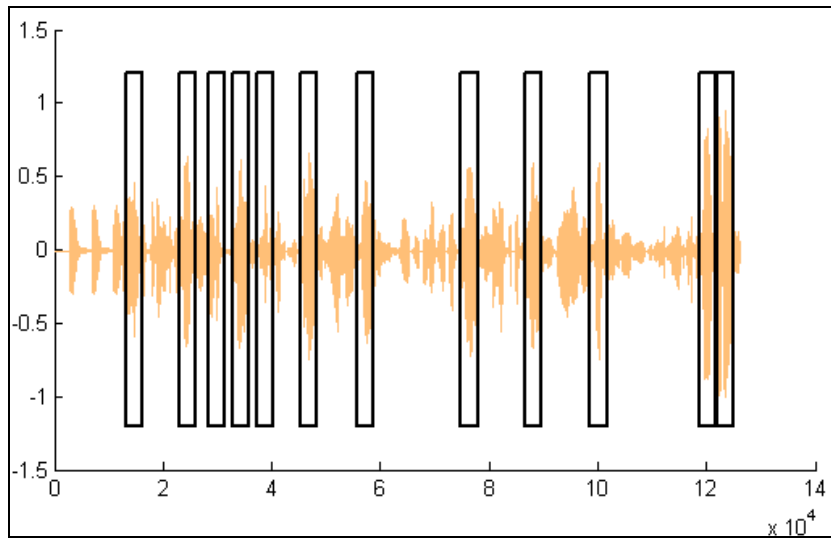

Figure 1: A test audio CAPTCHA with the fixed-size segments containing the highest energy peaks highlighted.

The information provided in the manual transcriptions of the audio CAPTCHAs contains a list of the time intervals within which words are spoken. However, these intervals are of variable size and the word might be spoken anywhere within this interval. To provide fixed-size segments for training, we developed the following heuristic. First, divide each file into variable-size segments using the time intervals provided and label each segment accordingly. Then, within each segment, detect the highest energy peak and return its fixed-size neighborhood labeled with the current segment's label. This heuristic achieved nearly perfect labeling accuracy for the training set. Rare mistakes occurred when the highest energy peak of a digit or letter segment corresponded to noise rather than to a digit or letter.

To summarize this subsection, an audio file is transformed into a set of fixed-size segments labeled as noise, a digit between 0 and 9, or a letter between a and z. These segments are then used for training. Classifiers are trained for one type of CAPTCHA at a time.

## 4 Classifier construction

From the training data we extracted five sets of features using twelve MFCCs and twelfth-

order spectral (SPEC) and cepstral (CEPS) coefficients from PLP and RASTA-PLP. The Matlab functions for extracting these features were provided online at [7] and as part of the Voicebox package. We use AdaBoost, SVM, and $k$-NN algorithms to implement automated digit and letter recognition. We detail our implementation of each algorithm in the following subsections.

## 4.1    AdaBoost

Using decision stumps as weak classifiers for AdaBoost, anywhere from 11 to 37 ensemble classifiers are built. The number of classifiers built depends on which type of CAPTCHA we are solving. Each classifier trains on all the segments associated with that type of CAPTCHA, and for the purpose of building a single classifier, segments are labeled by either -1 (negative example) or +1 (positive example). Using cross-validation, we choose to use 50 iterations for our AdaBoost algorithm. A segment can then be classified as a particular letter, digit, or noise according to the ensemble classifier that outputs the number closest to 1.

## 4.2    Support vector machine

To conduct digit recognition with SVM, we used the C++ implementations of libSVM [8] version 2.85 with C-SMV and RBF kernel. First, all feature values are scaled to the range of -1 to 1 as suggested by [8]. The scale parameters are stored so that test samples can be scaled accordingly. Then, a single multiclass classifier is created for each set of features using all the segments for a particular type of CAPTCHA. We use cross-validation and grid search to discover the optimal slack penalty ($C=32$) and kernel parameter ($\gamma=0.011$).

## 4.3    $k$-nearest neighbor ($k$-NN)

We use $k$-NN as our final method for classifying digits. For each type of CAPTCHA, five different classifiers are created by using all of the training data and the five sets of features associated with that particular type of CAPTCHA. Again we use cross-validation to discover the optimal parameter, in this case $k=1$. We use Euclidian distance as our distance metric.

# 5    Assessment of current audio CAPTCHAs

Our method for solving CAPTCHAs iteratively extracts an audio segment from a CAPTCHA, inputs the segment to one of our digit or letter recognizers, and outputs the label for that segment. We continue this process until the maximum solution size is reached or there are no unlabeled segments left. Some of the CAPTCHAs we evaluated have solutions that vary in length. Our method ensures that we get solutions of varying length that are never longer than the maximum solution length. A segment to be classified is identified by taking the neighborhood of the highest energy peak of an as yet unlabeled part of the CAPTCHA.

Once a prediction of the solution to the CAPTCHA is computed, it is compared to the true solution. Given that at least one of the audio CAPTCHAs allows users to make a mistake in one of the digits (e.g., reCAPTCHA), we compute the pass rate for each of the different types of CAPTCHAs with all of the following conditions:

- The prediction matches the true solution exactly.

- Inserting one digit into the prediction would make it match the solution exactly.

- Replacing one digit in the prediction would make it match the solution exactly.

- Removing one digit from the prediction would make it match the solution exactly.

However, since we are only sure that these conditions apply to reCAPTCHA audio CAPTCHAs, we also calculate the percentage of exact solution matches in our results for each type of audio CAPTCHA. These results are described in the following subsections.

## 5.1    Google

Google audio CAPTCHAs consist of one speaker saying random digits 0-9, the phrase "once again," followed by the exact same recorded sequence of digits originally presented.

The background noise consists of human voices speaking backwards at varying volumes. A solution can range in length from five to eight words. We set our classifier to find the 12 loudest segments and classify these segments as digits or noise. Because the phrase "once again" marks the halfway point of the CAPTCHA, we preprocessed the audio to only serve this half of the CAPTCHA to our classifiers. It is important to note, however, that the classifiers were always able to identify the segment containing "once again," and these segments were identified before all other segments. Therefore, if necessary, we could have had our system cut the file in half after first labeling this segment.

For AdaBoost, we create 12 classifiers: one classifier for each digit, one for noise, and one for the phrase "once again." Our results (Table 1) show that at best we achieved a 90% pass rate using the "one mistake" passing conditions and a 66% exact solution match rate. Using SVM and the "one mistake" passing conditions, at best we achieve a 92% pass rate and a 67% exact solution match. For *k*-NN, the "one mistake" pass rate is 62% and the exact solution match rate is 26%.

Table 1: Google audio CAPTCHA results: Maximum 67% accuracy was achieved by SVM.

| | | Classifiers Used | | | | | |
| --- | --- | --- | --- | --- | --- | --- | --- |
| | | AdaBoost | | SVM | | *k*-NN | |
| | | One mistake | exact match | one mistake | exact match | one mistake | exact match |
| **Features Used** | MFCC | 88% | 61% | 92% | **67%** | 30% | 1% |
| | PLP-SPEC | 90% | 66% | 90% | **67%** | 60% | 26% |
| | PLP-CEPS | 90% | 66% | 92% | **67%** | 62% | 23% |
| | RASTA-PLP-SPEC | 88% | 48% | 90% | 61% | 29% | 1% |
| | RASTA-PLP-CEPS | 90% | 63% | 92% | **67%** | 33% | 2% |

## 5.2   Digg

Digg CAPTCHAs also consist of one speaker, in this case saying a random combination of letters and digits. The background noise consists of static or what sounds like trickling water and is not continuous throughout the entire file. We noticed in our training data that the following characters were never present in a solution: 0, 1, 2, 5, 7, 9, i, o, z. Since the Digg audio CAPTCHA is also the verbal transcription of the visual CAPTCHA, we believe that these characters are excluded to avoid confusion between digits and letters that are similar in appearance. The solution length varies between three and six words. Using AdaBoost, we create 28 classifiers: one classifier for each digit or letter that appears in our training data and one classifier for noise. Perhaps because we had fewer segments to train with and there was a far higher proportion of noise segments, AdaBoost failed to produce any correct solutions. We believe that the overwhelming number of negative training examples versus the small number of positive training samples used to create each decision stump severely affected AdaBoost's ability to classify audio segments correctly.

A histogram of the training samples is provided in Figure 2 to illustrate the amount of training data available for each character. When using SVM, the best feature set passed with 96% using "one mistake" passing conditions and passed with 71% when matching the solution exactly. For *k*-NN, the best feature set produced a 90% "one mistake" pass rate and a 49% exact solution match. Full results can be found in Table 2.

Table 2: Digg audio CAPTCHA results: Maximum 71% accuracy was achieved by SVM.

| | | Classifiers Used | | | | | |
| --- | --- | --- | --- | --- | --- | --- | --- |
| | | AdaBoost | | SVM | | k-NN | |
| | | one mistake | exact match | one mistake | exact match | one mistake | exact match |
| **Features Used** | MFCC | - | - | 96% | **71%** | 89% | 49% |
| | PLP-SPEC | - | - | 94% | 65% | 90% | 47% |
| | PLP-CEPS | - | - | 96% | **71%** | 64% | 17% |
| | RASTA-PLP-SPEC | - | - | 17% | 3% | 67% | 17% |
| | RASTA-PLP-CEPS | - | - | 96% | **71%** | 82% | 34% |

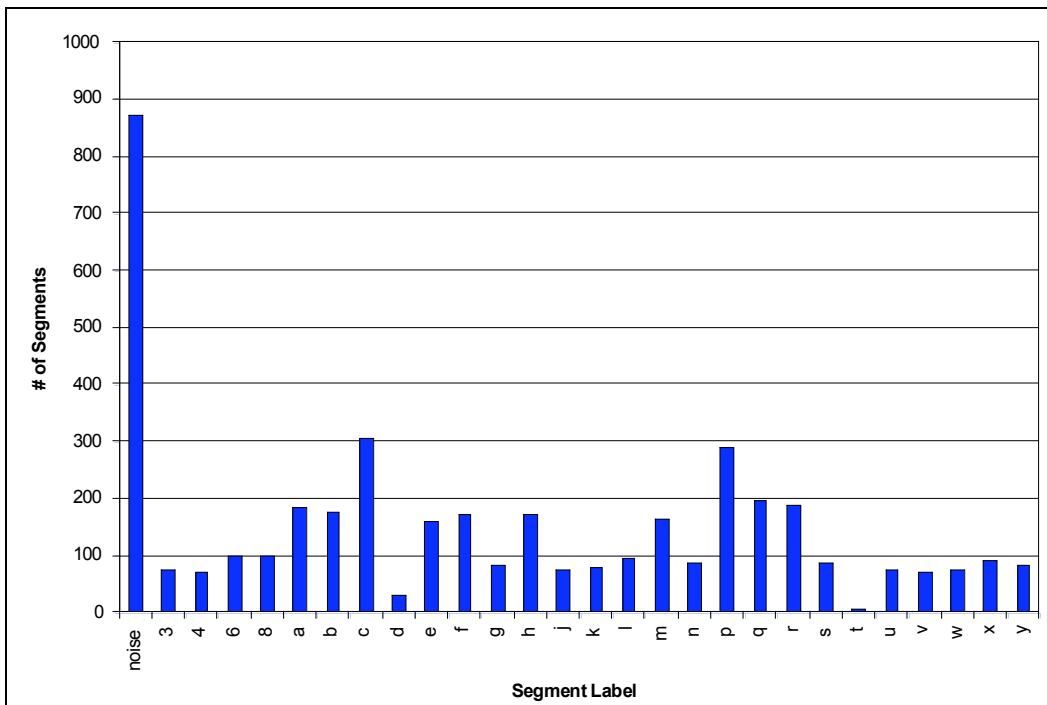

Figure 2: Digg CAPTCHA training data distribution.

## 5.3    reCAPTCHA

The older version of reCAPTCHA's audio CAPTCHAs we tested consist of several speakers who speak random digits. The background noise consists of human voices speaking backwards at varying volumes. The solution is always eight digits long. For AdaBoost, we create 11 classifiers: one classifier for each digit and one classifier for noise. Because we

know that the reCAPTCHA passing conditions are the "one mistake" passing conditions, SVM produces our best pass rate of 58%. Our best exact match rate is 45% (Table 3).

Table 3: reCAPTCHA audio CAPTCHA results: Maximum 45% accuracy was achieved by SVM.

| | | Classifiers Used | | | | | |
| | | AdaBoost | | SVM | | k-NN | |
| | | one mistake | exact match | one mistake | exact match | one mistake | exact match |
|---|---|---|---|---|---|---|---|
| Features Used | MFCC | 18% | 6% | 56% | 43% | 22% | 11% |
| | PLP-SPEC | 27% | 10% | 58% | 39% | 43% | 25% |
| | PLP-CEPS | 23% | 10% | 56% | **45%** | 29% | 14% |
| | RASTA-PLP-SPEC | 9% | 3% | 36% | 18% | 24% | 4% |
| | RASTA-PLP-CEPS | 9% | 3% | 46% | 30% | 32% | 12% |

# 6    Properties of weak versus strong CAPTCHAs

From our results, we note that the easiest CAPTCHAs to break were from Digg. Google had the next strongest CAPTCHAs followed by the strongest from reCAPTCHA. Although the Digg CAPTCHAs have the largest vocabulary, giving us less training data per label, the same woman recorded them all. More importantly, the same type of noise is used throughout the entire CAPTCHA. The noise sounds like running water and static which sounds very different from the human voice and does not produce the same energy spikes needed to locate segments, therefore making segmentation quite easy. The CAPTCHAs from Google and reCAPTCHA used other human voices for background noise, making segmentation much more difficult. Although Google used a smaller vocabulary than Digg and also only used one speaker, Google's background noise made the CAPTCHA more difficult to solve. After listening to a few of Google's CAPTCHAs, we noticed that although the background noise consisted of human voices, the same background noise was repeated. reCAPTCHA had similar noise to Google, but they had a larger selection of noise thus making it harder to learn. reCAPTCHA also has the longest solution length making it more difficult to get perfectly correct. Finally, reCAPTCHA used many different speakers causing it to be the strongest CAPTCHA of the three we tested. In conclusion, an audio CAPTCHA that consists of a finite vocabulary and background noise should have multiple speakers and noise similar to the speakers.

# 7    Recommendations for creating stronger audio CAPTCHAs

Due to our success in solving audio CAPTCHAs, we have decided to start developing new audio CAPTCHAs that our methods, and machine learning methods in general, will be less likely to solve. From our experiments, we note that CAPTCHAs containing longer solutions and multiple speakers tend to be more difficult to solve. Also, because our methods depend on the amount of training data we have, having a large vocabulary would make it more difficult to collect enough training data. Already since obtaining these results, reCAPTCHA.net has updated their audio CAPTCHA to contain more distortions and a

larger vocabulary: the digits 0 through 99. In designing a new audio CAPTCHA we are also concerned with the human pass rate. The current human pass rate for the reCAPTCHA audio CAPTCHAs is only 70%. To develop an audio CAPTCHA with an improved human pass rate, we plan to take advantage of the human mind's ability to understand distorted audio through context clues. By listening to a phrase instead of to random isolated words, humans are better able to decipher distorted utterances because they are familiar with the phrase or can use contextual clues to decipher the distorted audio. Using this idea, the audio for our new audio CAPTCHA will be taken from old-time radio programs in which the poor quality of the audio makes transcription by ASR systems difficult. Users will be presented with an audio clip consisting of a 4-6 word phrase. Half of the CAPTCHA consists of words, which validate a user to be human, while the other half of the words need to be transcribed. This is the same idea behind the visual reCAPTCHA that is currently digitizing text on which OCR fails. We expect that this new audio CAPTCHA will be more secure than the current version and easier for humans to pass. Initial experiments using this idea show this to be true [9].

# 8    Conclusion

We have succeeded in "breaking" three different types of widely used audio CAPTCHAs, even though these were developed with the purpose of defeating attacks by machine learning techniques. We believe our results can be improved by selecting optimal segment sizes, but that is unnecessary given our already high success rate. For our experiments, segment sizes were not chosen in a special way; occasionally yielding results in which a segment only contained half of a word, causing our prediction to contain that particular word twice. We also believe that the AdaBoost results can be improved, particularly for the Digg audio CAPTCHAs, by ensuring that the number of negative training samples is closer to the number of positive training samples. We have shown that our approach is successful and can be used with many different audio CAPTCHAs that contain small finite vocabularies.

## Acknowledgments

This work was partially supported by generous gifts from the Heinz Endowment, by an equipment grant from Intel Corporation, and by the Army Research Office through grant number DAAD19-02-1-0389 to CyLab at Carnegie Mellon University. Luis von Ahn was partially supported by a Microsoft Research New Faculty Fellowship and a MacArthur Fellowship. Jennifer Tam was partially supported by a Google Anita Borg Scholarship.

## References

[1] L. von Ahn, M. Blum, and J. Langford. "Telling Humans and Computers Apart Automatically," *Communications of the ACM,* vol. 47, no. 2, pp. 57-60, Feb. 2004.

[2] G. Mori and J. Malik. "Recognizing Objects in Adversarial Clutter: Breaking a Visual CAPTCHA," In *Computer Vision and Pattern Recognition CVPR'03*, June 2003.

[3] K. Chellapilla, and P. Simard, "Using Machine Learning to Break Visual Human Interaction Proofs (HIPs)," *Advances in Neural Information Processing Systems 17, Neural Information Processing Systems* (NIPS'2004), MIT Press.

[4] H. Hermansky, "Perceptual Linear Predictive (PLP) Analysis of Speech," *J. Acoust. Soc. Am.,* vol. 87, no. 4, pp. 1738-1752, Apr. 1990.

[5] H. Hermansky, N. Morgan, A. Bayya, and P. Kohn. "RASTA-PLP Speech Analysis Technique," In *Proc. IEEE Int'l Conf. Acoustics, Speech & Signal Processing,* vol. 1, pp. 121-124, San Francisco, 1992.

[6] R. Santamarta. "Breaking Gmail's Audio Captcha," http://blog.wintercore.com/?p=11, 2008.

[7] D. Ellis. "PLP and RASTA (and MFCC, and inversion) in Matlab using melfcc.m and invmelfcc.m," http://www.ee.columbia.edu/~dpwe/resources/matlab/rastamat/, 2006.

[8] C. Chang and C. Lin. LIBSVM: a library for support vector machines, 2001. Software available at http://www.csie.ntu.edu.tw/~cjlin/libsvm

[9] A. Schlaikjer. "A Dual-Use Speech CAPTCHA: Aiding Visually Impaired Web Users while Providing Transcriptions of Audio Streams," *Technical Report CMU-LTI-07-014*, Carnegie Mellon University. November 2007.
